# A kernel method for multi-labelled classification

**André Elisseeff and Jason Weston**
BIOwulf Technologies, 305 Broadway, New York, NY 10007
{andre,jason}@barhilltechnologies.com

## Abstract

This article presents a Support Vector Machine (SVM) like learning system to handle multi-label problems. Such problems are usually decomposed into many two-class problems but the expressive power of such a system can be weak [5, 7]. We explore a new direct approach. It is based on a large margin ranking system that shares a lot of common properties with SVMs. We tested it on a Yeast gene functional classification problem with positive results.

## 1   Introduction

Many problems in Text Mining or Bioinformatics are multi-labelled. That is, each point in a learning set is associated to a set of labels. Consider for instance the classification task of determining the subjects of a document, or of relating one protein to its many effects on a cell. In either case, the learning task would be to output a set of labels whose size is not known in advance: one document can for instance be about food, meat and finance, although another one would concern only food and fat. Two-class and multi-class classification or ordinal regression problems can all be cast into multi-label ones. This makes the latter quite attractive but at the same time it gives a warning: their generality hides their difficulty to solve them. The number of publications is not going to contradict this statement: we are aware of only a few works about the subject [4, 5, 7] and they all concern text mining applications.

In Schapire and Singer's work about Boostexter, one of the only general purpose multi-label ranking systems [7], they observe that overfitting occurs on learning sets of relatively small size ($< 1000$). They conclude that controlling the complexity of the overall learning system is an important research goal. The aim of the current paper is to provide a way of controlling this complexity while having a small empirical error. For that purpose, we consider only architectures based on linear models and follow the same reasoning as for the definition of Support Vector Machines [1]. Defining a cost function (section 2) and margin for multi-label models, we focus our attention mainly on an approach based on a ranking method combined with a predictor of the size of the sets (section 3 and 4). Sections 5 and 6 present experiments on a toy problem and on a real dataset.

## 2   Cost functions

Let $\mathcal{X} = \mathbb{R}^d$ be a d-dimensional input space. We consider as an output space the space $\mathcal{Y}$ formed by all the sets of integer between 1 and $Q$ identified here as the labels of the

learning problem. Such an output space contains $2^Q$ elements and one output corresponds to one set of labels. The learning problem we are interested in is to find from a learning set $S = \{(x_1, \mathbf{Y}_1), \ldots, (x_m, \mathbf{Y}_m)\} \subset (\mathcal{X} \times \mathcal{Y})^m$, drawn identically and independently from an unknown distribution $D$, a function $f$ such that the following generalization error is as low as possible:

$$R(f) = E_{(x,\mathbf{Y}) \sim D}\left[c(f, x, \mathbf{Y})\right] \tag{1}$$

The function $c$ is a real-valued loss and can take different forms depending on how $f(x)$ is computed. Here, we consider only linear models. Given $Q$ vectors $w_1, \ldots, w_Q$ and $Q$ bias $b_1, \ldots, b_Q$, we follow two schemes:

*With the binary approach:* $f(x) = \text{sign}\left(\langle w_1, x \rangle + b_1, \ldots, \langle w_Q, x \rangle + b_Q\right)$, where the sign function applies component-wise. The value of $f(x)$ is a binary vector from which the set of labels can be retrieved easily by stating that label $k$ is in the set iff $\text{sign}(\langle w_k, x \rangle + b_k) > 0$. For example this can be achieved by using a SVM for each binary problem and applying the latter rule [4].

*With the ranking approach:* assume that $s(x)$, the size of the label set for the input $x$, is known. We define: $r_k(x) = \langle w_k, x \rangle + b_k$ and consider that a label $k$ is in the label set of $x$ iff $r_k(x)$ is among the largest $s(x)$ elements $(r_1(x), .., r_Q(x))$. The algorithm Boostexter [7] is an example of such a system. The ranking approach is analyzed more precisely in section 3.

We consider the same loss functions as in [7] for any multi-label system built from real functions $(f_1, .., f_Q)$. It includes the so-called *Hamming Loss* defined as

$$HL(f, x, \mathbf{Y}) = \frac{1}{Q}\left|f(x)\Delta\mathbf{Y}\right|$$

where $\Delta$ stands for the symmetric difference of sets. When $|\mathbf{Y}| = 1$ a multi-label system is in fact a multi-class one and the Hamming Loss is $\frac{2}{Q}$ times the loss of the usual classification loss. We also consider the *one-error*:

$$\text{1-err}(f, x, \mathbf{Y}) = \begin{cases} 0 & \text{if } \text{argmax}_k f_k(x) \in \mathbf{Y} \\ 1 & \text{otherwise} \end{cases}$$

which is exactly the same as the classification error for multi-class problems (it ignores the rankings apart from the highest ranked one and so does not address the quality of the other labels).

Other losses concern only ranking systems (a system that specifies a ranking but no set size predictor $s(x)$). Let us denote by $\bar{\mathbf{Y}}$ the complementary set of $\mathbf{Y}$ in $\{1, .., Q\}$. We define the *Ranking Loss* [7] to be:

$$RL(f, x, \mathbf{Y}) = \frac{1}{|\mathbf{Y}||\bar{\mathbf{Y}}|}\left|(i,j) \in \mathbf{Y} \times \bar{\mathbf{Y}} \text{ s.t. } r_i(x) \leq r_j(x)\right| \tag{2}$$

It represents the average fraction of pairs that are not correctly ordered. For ranking systems, this loss is natural and is related to the *precision* which is a common error measure in Information Retrieval:

$$\text{precision}(f, x, \mathbf{Y}) = \frac{1}{|\mathbf{Y}|}\sum_{k \in \mathbf{Y}} \frac{|\{l \in \mathbf{Y} \text{ s.t. } r_l(x) \geq r_k(x)\}|}{|\{l \in \{1, .., Q\} \text{ s.t. } r_l(x) \geq r_k(x)\}|}$$

from which a loss can be directly deduced. All these loss functions have been discussed in [7]. Good systems should have a high precision and a low Hamming or Ranking Loss. We do not consider the one-error to be a good loss for multi-label systems but we retain it because it was measured in [7].

For multi-label linear models, we need to define a way of minimizing the empirical error measured by the appropriate loss and at the same time to control the complexity of the resulting model. A direct method would be to use the binary approach and thus take the benefit of good two-class systems. However, as it has been raised in [5, 7], the binary approach does not take into account the correlation between labels and therefore does not capture the structure of some learning problems. We propose here to instead focus on the ranking approach. This will be done by introducing notions of margin and regularization as has been done for the two-class case in the definition of SVMs.

## 3  Ranking based system

Our goal is to define a linear model that minimizes the Ranking Loss while having a large margin. For systems that rank the values of $\langle w_k, x \rangle + b_k$, the decision boundaries for $x$ are defined by the hyperplanes whose equations are $\langle w_k - w_l, x \rangle + b_k - b_l = 0$, where $k$ belongs to the label sets of $x$ and $l$ does not. So, the margin of $(x, \mathbf{Y})$ can be expressed as:

$$\min_{k \in \mathbf{Y}, l \in \bar{\mathbf{Y}}} \frac{\langle w_k - w_l, x \rangle + b_k - b_l}{\|w_k - w_l\|}$$

It represents the signed $\ell_2$ distance of $x$ to the decision boundary. Considering that all the data in the learning set $S$ are well ranked, we can normalize the parameters $w_k$ such that:

$$\langle w_k - w_l, x \rangle + b_k - b_l \geq 1$$

with equality for some $x \in S$, and $(k, l) \in \mathbf{Y} \times \bar{\mathbf{Y}}$. Maximizing the margin on the whole learning set can then be done via the following problem:

$$\max_{w_j, j=1,..,Q} \quad \min_{(x,\mathbf{Y}) \in S} \min_{k \in \mathbf{Y}, l \in \bar{\mathbf{Y}}} \frac{1}{\|w_k - w_l\|^2} \tag{3}$$

$$\text{subject to:} \quad \langle w_k - w_l, x_i \rangle + b_k - b_l \geq 1, \ (k, l) \in \mathbf{Y}_i \times \bar{\mathbf{Y}}_i \tag{4}$$

In the case where the problem is not ill-conditioned (two labels are always co-occurring), the objective function can be replaced by: $\max_{w_j} \min_{k,l} \frac{1}{\|w_k - w_l\|^2} = \min_{w_j} \max_{k,l} \|w_k - w_l\|^2$. In order to get a simpler optimization procedure we approximate this maximum by the sum and, after some calculations (see [3] for details), we obtain:

$$\min_{w_j, j=1,..,Q} \quad \sum_{k=1}^{Q} \|w_k\|^2 \tag{5}$$

$$\text{subject to:} \quad \langle w_k - w_l, x_i \rangle + b_k - b_l \geq 1, \ (k, l) \in \mathbf{Y}_i \times \bar{\mathbf{Y}}_i \tag{6}$$

To generalize this problem in the case where the learning set can not be ranked exactly we follow the same reasoning as for the binary case: the ultimate goal would be to maximize the margin and at the same time to minimize the Ranking Loss. The latter can be expressed quite directly by extending the constraints of the previous problems. Indeed, if we have $\langle w_k - w_l, x_i \rangle + b_k - b_l \geq 1 - \xi_{ikl}$ for $(k, l) \in \mathbf{Y}_i \times \bar{\mathbf{Y}}_i$, then the Ranking Loss on the learning set $S$ is:

$$\frac{1}{m} \sum_{i=1}^{m} \frac{1}{|\mathbf{Y}_i||\bar{\mathbf{Y}}_i|} \sum_{k,l \in (\mathbf{Y}_i \times \bar{\mathbf{Y}}_i)} \theta(-1 + \xi_{ikl})$$

where $\theta$ is the Heaviside function. As for SVMs we approximate the functions $\theta(-1 + \xi_{ikl})$ by only $\xi_{ikl}$ and this gives the final quadratic optimization problem:

$$\min_{w_j, j=1,..,Q} \quad \sum_{k=1}^{Q} \|w_k\|^2 + C \sum_{i=1}^{m} \frac{1}{|\mathbf{Y}_i||\bar{\mathbf{Y}}_i|} \sum_{(k,l) \in \mathbf{Y}_i \times \bar{\mathbf{Y}}_i} \xi_{ikl} \tag{7}$$

$$\text{subject to:} \quad \langle w_k - w_l, x_i \rangle + b_k - b_l \geq 1 - \xi_{ikl}, \ (k, l) \in \mathbf{Y}_i \times \bar{\mathbf{Y}}_i \tag{8}$$

$$\xi_{ikl} \geq 0 \tag{9}$$

In the case where the label sets $\mathbf{Y}_i$ all have a size of 1 we find the same optimization problem as has been derived for multi-class Support Vector Machines [8]. For this reason, we call the solution of this problem a ranking Support Vector Machine (Rank-SVM). Another common property with SVM is the possibility to use kernels rather than linear dot products. This can be achieved by computing the dual of the former optimization problem. We refer the reader to [3] for the dual formluation and to [2] and references therein for more information about kernels and SVMs.

Solving a constrained quadratic problem like those we just introduced requires an amount of memory that is quadratic in terms of the learning set size and it is generally solved in $O(m^3)$ computational steps where we have put into the $O$ the number of labels. Such a complexity is too high to apply these methods in many real datasets. To circumvent this limitation, we propose to use a linearization method in conjunction with a predictor-corrector logarithmic barrier procedure. Details are described in [3] with all the calculations relative to the implementation. The memory cost of the method then becomes $O(mQQ_{max})$ where $Q_{max} = \max_i |\mathbf{Y}_i|$ is the maximum number of labels. In many applications $Q$ is much larger than $Q_{max}$. The time cost of each iteration is $O(m^2Q)$.

## 4   Set size prediction

So far we have only developed ranking systems. To obtain a complete multi-label system we need to design a set size predictor $s(x)$. A natural way of doing this is to look for inspiration from the binary approach. The latter can indeed be interpreted as a ranking system whose ranks are derived from the real values $(f_1, .., f_Q)$. The predictor of the set size is then quite simple: $s(x) = |\{f_k(x) > 0\}|$ is the number of $f_k$ that are greater than 0. The function $s(x)$ is computed from a threshold value that differentiates labels in the target set from others. For the ranking system introduced in the previous section we generalize this idea by designing a function $s(x) = |\{f_k(x) > t(x)\}|$. The remaining problem now is to choose $t(x)$ which is done by solving a learning problem. The training data are composed by the $(f_1(x_i), .., f_Q(x_i))$ given by the ranking system, and by the target values defined by:

$$t(x_i) = \operatorname{argmin}_t \left|\{k \in \mathbf{Y} \text{ s.t. } f_k(x_i) \leq t\}\right| + \left|\{k \in \bar{\mathbf{Y}} \text{ s.t. } f_k(x_i) \geq t\}\right|$$

When the minimum is not unique and the optimal values are a segment, we choose the middle of this segment. We refer to this method of predicting the set size as the *threshold based* method. In the following, we have used linear least squares, and we applied it not only to Rank-SVM but also to Boostexter in order to transform these algorithms from ranking methods to multi-label ones.

Note that we could have followed a much simpler scheme to build the function $s(x)$. A naive method would be to consider the set size prediction as a regression problem on the original training data with the targets $(|\mathbf{Y}_i|)_{i=1,..,m}$ and to use any regression learning system. This however does not provide a satisfactory solution mainly because it does not take into account how the ranking is performed. In particular, when there are some errors in the ranking, it does not learn how to compensate these errors although the threshold based approach tries to learn the best threshold with respect to these errors.

## 5   Toy problem

As previously noticed the binary approach is not appropriate for problems where correlation between labels exist. To illustrate this point consider figure 2. There are only three labels. One of them (label 1) is present for all points in the learning set. The binary approach leads to a system that will fail to separate, for instance, points with label 3 from points of

label sets not containing 3, that is, on points of label 1 and 2. We see then that the expressible power of a binary system can be quite low when simple configurations occur. If we consider the ranking approach, one can imagine the following solution: $w_1 = 0$, $b_1 = \infty$, $(w_2, b_2)$ is the hyperplane separating class 2 from class 3, and $(w_3, b_3) = -(w_2, b_2)$. By taking the number of labels at point $x$ to be $s(x) = \langle w, x \rangle + b$ where $w = (-1, 1)$ and $b = 0$, we have a simple multi-label system that separates all the regions exactly.

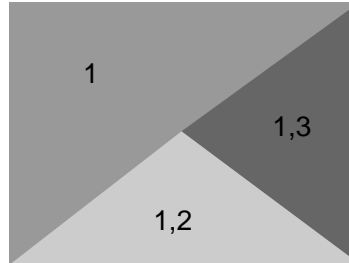

Figure 2: Three labels and three regions in the input space. The upper left region is labelled with 1. The bottom right region is partitioned into two sub-regions with labels 1, 2 or 1, 3.

To make this point more concrete we sampled 50 points uniformly on $[0, 1]^2$ and solved all optimization problems with $C = \infty$. On the learning set the Hamming Loss for the binary approach was $0.08$ although for the direct approach it was $0$ as expected.

## 6 Experiments on real data

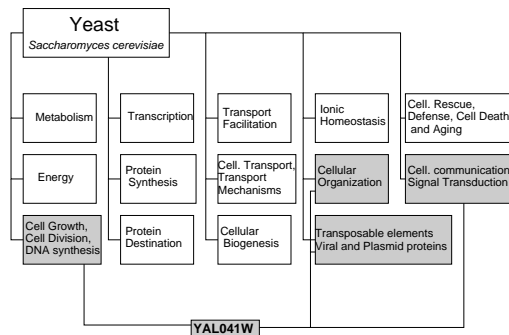

Figure 3: First level of the hierarchy of the gene functional classes. There are 14 groups. One gene, for instance the gene YAL041W can belong to different groups (shaded in grey on the figure).

The Yeast dataset is formed by micro-array expression data and phylogenetic profiles with 1500 genes in the learning set and 917 in the test set. The input dimension is 103. Each gene is associated with a set of functional classes whose maximum size can be potentially more than 190. This dataset has already been analyzed with a two-class approach [6] and is known to be difficult. In order to make it easier, we used the known structure of the functional classes. The whole set of classes is indeed structured in a tree whose leaves are the functional categories (see `http://mips.gsf.de/proj/yeast/catalogues/funcat/` for more details). Given a gene, knowing which edge to take from one level to another leads directly to a leaf and

thus to a functional class. Here we try to predict which edge to take from the root to the first level of the tree (see figure 3).

Since one gene can have many functional classes this is a multi-label problem: one gene is associated to different edges. We then have $Q = 14$ and the average number of labels for all genes in the learning set is $4.2 \pm 1.6$. We assessed the quality of our method from two perspectives. First as a ranking system with the Ranking Loss and the precision. In that case, for the binary approach, the real outputs of the two-class SVMs were used as ranking values. Second, the methods were compared as multi-label systems using the Hamming Loss. We computed the latter for the binary approach used in conjunction with SVMs, for the Rank-SVM and for Boostexter. To measure the Hamming Loss with Boostexter we used a threshold based $s(x)$ function in combination with the ranking given by the algorithm.

| | Rank-SVM | | | | Binary-SVM | | | |
|---|---|---|---|---|---|---|---|---|
| degree | 2 | 3 | 4 | 5 | 2 | 3 | 4 | 5 |
| Precision | **0.703** | **0.740** | **0.746** | **0.762** | 0.692 | 0.721 | 0.714 | 0.753 |
| Ranking Loss | **0.227** | **0.191** | **0.190** | **0.175** | 0.241 | 0.212 | 0.196 | 0.184 |
| Hamming Loss | **0.238** | **0.217** | **0.209** | **0.201** | 0.247 | 0.224 | 0.211 | 0.207 |
| one-error | **0.334** | **0.262** | **0.255** | **0.232** | 0.341 | 0.306 | 0.267 | 0.250 |

Figure 4: Polynomials of degree 2-5. Loss functions for the rank-SVM and the binary approach based on two-class SVMs. Considering the size of the problem, two values different from less than $0.01$ are not significantly different. Bold values represent superior performance comparing classifiers with the same kernel.

For rank-SVMs and for two-class SVMs in the binary approach we choose polynomial kernels of degrees two to nine (experiments on two-class problems using the Yeast data in [6] already showed that polynomial kernels were appropriate for this task). Boostexter was used with the standard stump weak learner and was stopped after 1000 iterations. Results are reported in tables 4, 5 and 6.

| | Rank-SVM | | | | Binary-SVM | | | |
|---|---|---|---|---|---|---|---|---|
| degree | 6 | 7 | 8 | 9 | 6 | 7 | 8 | 9 |
| Precision | **0.765** | **0.770** | **0.773** | 0.769 | 0.760 | 0.765 | 0.770 | 0.769 |
| Ranking Loss | **0.170** | **0.166** | **0.163** | **0.163** | 0.176 | 0.170 | 0.165 | 0.164 |
| Hamming Loss | **0.199** | **0.198** | 0.196 | 0.197 | 0.200 | 0.199 | **0.195** | **0.195** |
| one-error | 0.232 | **0.223** | **0.217** | **0.225** | 0.232 | 0.227 | 0.218 | 0.226 |

Figure 5: Polynomials of degree 6-9. Loss functions for the rank-SVM and the binary approach based on two-class SVMs. Considering the size of the problem, two values different from less than $0.01$ are not significantly different. Bold values represent superior performance comparing classifiers with the same kernel.

| | Boostexter (1000 iterations) |
|---|---|
| Precision | 0.717 |
| Ranking Loss | 0.298 |
| Hamming Loss | 0.237 |
| one-error | 0.302 |

Figure 6: Loss functions for Boostexter. Note that these results are worse than with the binary approach or with rank-SVM.

Note that Boostexter performs quite poorly on this dataset compared to SVM-based approaches. This may be due to the simple decision function realized by Boostexter. One

of the main advantages of the SVM-based approaches is the ability to incorporate priori knowledge into the kernel and control complexity via the kernel and regularization. We believe this may also be possible with Boostexter but we are not aware of any work in this area.

To compare the binary and the rank-SVM we put in bold the best results for each kernel. For all kernels and for almost all losses, the combination ranking based SVM approach is better than the binary one. In terms of the Ranking Loss, the difference is significantly in favor of the rank-SVM. It is consistent with the fact that this system tends to minimize this particular loss function. It is worth noticing that when the kernel becomes more and more complex the difference between rank-SVM and the binary method disappears.

## 7 Discussion and conclusion

In this paper we have defined a whole system to deal with multi-label problems. The main contribution is the definition of a ranking based SVM that extends the use of the latter to many problems in the area of Bioinformatics and Text Mining.

We have seen on complex, real data that rank-SVMs lead to better performance than Boostexter and the binary approach. On its own this could be interpreted as a sufficient argument to motivate the use of such a system. However, we can also extend the rank-SVM system to perform feature selection on ranking problems [3] . This application can be very useful in the field of bioinformatics as one is often interested in interpretability of a multi-label decision rule. For example one could be interested in a small set of genes which is discriminative in a multi-condition physical disorder.

We have presented only first experiments using multi-labelled systems applied to Bioinformatics. Our future work is to conduct more investigations in this area.

## References

[1] B. Boser, I. Guyon, and V. Vapnik. A training algorithm for optimal margin classifiers. In *Fifth Annual Workshop on Computational Learning Theory*, pages 144–152, Pittsburgh, 1992. ACM.

[2] N. Cristianini and J. Shawe-Taylor. *Introduction to Support Vector Machines*. Cambridge University Press, 2000.

[3] André Elisseeff and Jason Weston. Kernel methods for multi-labelled classification and categorical regression problems. Technical report, BIOwulf Technologies, 2001. http://www.bht-labs.com/public/.

[4] T. Joachims. Text categorization with support vector machines: learning with many relevant features. In Claire Nédellec and Céline Rouveirol, editors, *Proceedings of ECML-98, 10th European Conference on Machine Learning*, number 1398, pages 137–142, Chemnitz, DE, 1998. Springer Verlag, Heidelberg, DE.

[5] A. McCallum. Multi-label text classification with a mixture model trained by em. *AAAI'99 Workshop on Text Learning.*, 1999.

[6] P. Pavlidis, J. Weston, J. Cai, and W.N. Grundy. Combining microarray expression data and phylogenetic profiles to learn functional categories using support vector machines. In *RECOMB*, pages 242–248, 2001.

[7] R.E. Schapire and Y. Singer. Boostexter: A boosting-based system for text categorization. *Machine Learning*, 39(2/3):135–168, 2000.

[8] J. Weston and C. Watkins. Multi-class support vector machines. Technical Report 98-04, Royal Holloway, University of London, 1998.
